# Bootstrapping Apprenticeship Learning

**Abdeslam Boularias**
Department of Empirical Inference
Max-Planck Institute for Biological Cybernetics
72076 Tübingen, Germany
abdeslam.boularias@tuebingen.mpg.de

**Brahim Chaib-Draa**
Department of Computer Science
Laval University
Quebec G1V 0A6, Canada
chaib@damas.ift.ulaval.ca

## Abstract

We consider the problem of apprenticeship learning where the examples, demonstrated by an expert, cover only a small part of a large state space. Inverse Reinforcement Learning (IRL) provides an efficient tool for generalizing the demonstration, based on the assumption that the expert is maximizing a utility function that is a linear combination of state-action features. Most IRL algorithms use a simple Monte Carlo estimation to approximate the expected feature counts under the expert's policy. In this paper, we show that the quality of the learned policies is highly sensitive to the error in estimating the feature counts. To reduce this error, we introduce a novel approach for bootstrapping the demonstration by assuming that: $(i)$, the expert is (near-)optimal, and $(ii)$, the dynamics of the system is known. Empirical results on gridworlds and car racing problems show that our approach is able to learn good policies from a small number of demonstrations.

## 1  Introduction

Modern robots are designed to perform complicated planning and control tasks, such as manipulating objects, navigating in outdoor environments, and driving in urban settings. Unfortunately, manually programming these tasks is almost infeasible in practice due to their high number of states. Markov Decision Processes (MDPs) provide an efficient tool for handling such tasks with a little help from an expert. The expert's help consists in simply specifying a reward function. However, in many practical problems, even specifying a reward function is not easy. In fact, it is often easier to demonstrate examples of a desired behavior than to define a reward function (Ng & Russell, 2000).

Learning policies from demonstration, a.k.a. apprenticeship learning, is a technique that has been widely used in robotics. An efficient approach to apprenticeship learning, known as Inverse Reinforcement Learning (IRL) (Ng & Russell, 2000; Abbeel & Ng, 2004), consists in recovering a reward function under which the policy demonstrated by an expert is near-optimal, rather than directly mimicking the expert's actions. The learned reward is then used for finding an optimal policy. Consequently, the expert's actions can be predicted in states that have not been encountered during the demonstration. Unfortunately, as already pointed by Abbeel & Ng (2004), recovering a reward function is an ill-posed problem. In fact, the expert's policy can be optimal under an infinite number of reward functions. Most of the work on apprenticeship learning via IRL focused on solving this particular problem by using different types of regularization and loss cost functions (Ratliff et al., 2006; Ramachandran & Amir, 2007; Syed & Schapire, 2008; Syed et al., 2008).

In this paper, we focus on another important problem occurring in IRL. IRL-based algorithms rely on the assumption that the reward function is a linear combination of state-action features. Therefore, the value function of any policy is a linear combination of the expected discounted frequency (count) of encountering each state-action feature. In particular, the value function of the expert's policy is approximated by a linear combination of the empirical averages of the features, estimated from the demonstration (the trajectories). In practice, this method works efficiently only if the number

of examples is sufficiently large to cover all the states, or the dynamics of the system is nearly deterministic. For the tasks related to systems with a stochastic dynamics and a limited number of available examples, we propose an alternative method for approximating the expected frequencies of the features under the expert's policy. Our approach takes advantage of the fact that the expert's partially demonstrated policy is near-optimal, and generalizes the expert's policy beyond the states that appeared in the demonstration. We show that this technique can be efficiently used to improve the performance of two known IRL algorithms, namely Maximum Margin Planning (MMP) (Ratliff et al., 2006), and Linear Programming Apprenticeship Learning (LPAL) (Syed et al., 2008).

## 2   Preliminaries

Formally, a finite-state Markov Decision Process (MDP) is a tuple $(\mathcal{S}, \mathcal{A}, \{T^a\}, R, \alpha, \gamma)$, where: $\mathcal{S}$ is a set of states, $\mathcal{A}$ is a set of actions, $T^a$ is a transition matrix defined as $\forall s, s' \in \mathcal{S}, a \in \mathcal{A}$ : $T^a(s, s') = Pr(s_{t+1} = s'|s_t = s, a_t = a)$, $R$ is a reward function ($R(s, a)$ is the reward associated with the execution of action $a$ in state $s$), $\alpha$ is the initial state distribution, and $\gamma$ is a discount factor. We denote by MDP\R a Markov Decision Process without a reward function, i.e. a tuple $(\mathcal{S}, \mathcal{A}, \{T^a\}, \alpha, \gamma)$. We assume that the reward function $R$ is given by a linear combination of $k$ feature vectors $f_i$ with weights $w_i$: $\forall s \in \mathcal{S}, \forall a \in \mathcal{A} : R(s, a) = \sum_{i=0}^{k} w_i f_i(s, a)$. A deterministic policy $\pi$ is a function that returns an action $\pi(s)$ for each state $s$. A stochastic policy $\pi$ is a probability distribution on the action to be executed in each state, defined as $\pi(s, a) = Pr(a_t = a|s_t = s)$. The value $V(\pi)$ of a policy $\pi$ is the expected sum of rewards that will be received if policy $\pi$ will be followed, i.e. $V(\pi) = E[\sum_{t=0}^{\infty} \gamma^t R(s_t, a_t)|\alpha, \pi, T]$. An optimal policy $\pi$ is one satisfying $\pi = \arg\max_\pi V(\pi)$. The occupancy $\mu_\pi$ of a policy $\pi$ is the discounted state-action visit distribution, defined as: $\mu_\pi(s, a) = E[\sum_{t=0}^{\infty} \gamma^t \delta_{s_t,s} \delta_{a_t,a}|\alpha, \pi, T]$ where $\delta$ is the Kronecker delta. We also use $\mu_\pi(s)$ to denote $\sum_a \mu_\pi(s, a)$. The following linear constraints, known as Bellman-flow constraints, are necessary and sufficient for defining an occupancy measure of a policy:

$$\{\big(\mu_\pi(s) = \alpha(s) + \gamma \sum_{s' \in \mathcal{S}} \sum_{a \in \mathcal{A}} \mu_\pi(s', a) T^a(s', s)\big), \big(\sum_{a \in \mathcal{A}} \mu_\pi(s, a) = \mu_\pi(s)\big), \big(\mu_\pi(s, a) \geqslant 0\big)\} \quad (1)$$

A policy $\pi$ is well-defined by its occupancy measure $\mu_\pi$, one can interchangeably use $\pi$ and $\mu_\pi$ to denote a policy. The set of feasible occupancy measures is denoted by $\mathcal{G}$. The frequency of a feature $f_i$ for a policy $\pi$ is given by $v_{i,\pi} = F(i, .)\mu_\pi$, where $F$ is a $k$ by $|\mathcal{S}||\mathcal{A}|$ feature matrix, such that $F(i, (s, a)) = f_i(s, a)$. Using this definition, the value of a policy $\pi$ can be written as a linear function of the frequencies: $V(\pi) = w^T F \mu_\pi = w^T v_\pi$, where $v_\pi$ is the vector of $v_{i,\pi}$. Therefore, the value of a policy is completely determined by the frequencies (or counts) of the features $f_i$.

## 3   Apprenticeship Learning

### 3.1   Overview

The aim of apprenticeship learning is to find a policy $\pi$ that is at least as good as a policy $\pi^E$ demonstrated by an expert, i.e. $V(\pi) \geqslant V(\pi^E)$. The value functions of $\pi$ and $\pi^E$ cannot be directly compared, unless a reward function is provided. To solve this problem, Ng & Russell (2000) proposed to first learn a reward function, assuming that the expert is optimal, and then use it to recover the expert's complete policy. However, the problem of learning a reward function given an optimal policy is ill-posed (Abbeel & Ng, 2004). In fact, a large class of reward functions, including all constant functions for instance, may lead to the same optimal policy. To overcome this problem, Abbeel & Ng (2004) did not consider recovering a reward function, instead, their algorithm returns a policy $\pi$ with a bounded loss in the value function, i.e. $\| V(\pi) - V(\pi^E) \| \leqslant \epsilon$, where the value is calculated by using the worst-case reward function. This property is derived from the fact that when the frequencies of the features under two policies match, the cumulative rewards of the two policies match as well, assuming that the reward is a linear function of these features. In the next two subsections, we briefly describe two algorithms for apprenticeship learning via IRL. The first one, known as Maximum Margin Planning (MMP) (Ratliff et al., 2006), is a robust algorithm based on learning a reward function under which the expert's demonstrated actions are optimal. The second one, known as Linear Programming Apprenticeship Learning (LPAL) Syed et al. (2008), is a fast algorithm that directly returns a policy with a bounded loss in the value.

## 3.2 Maximum Margin Planning

Maximum Margin Planning (MMP) returns a vector of reward weights $w$, such that the value of the expert's policy $w^T F \mu_{\pi^E}$ is higher than the value of an alternative policy $w^T F \mu_\pi$ by a margin that scales with the number of expert's actions that are different from the actions of the alternative policy. This criterion is explicitly specified in the cost function minimized by the algorithm:

$$c_q(w) = \left( \max_{\mu \in \mathcal{G}} (w^T F + l)\mu - w^T F \mu_{\pi^E} \right)^q + \frac{\lambda}{2} \parallel w \parallel^2 \tag{2}$$

where $q \in \{1, 2\}$ defines the slack penalization, $\lambda$ is a regularization parameter, and $l$ is a deviation cost vector, that can be defined as: $l(s, a) = 1 - \pi^E(s, a)$. A policy maximizing the cost-augmented reward vector $(w^T F + l)$ is almost completely different from $\pi^E$, since an additional reward $l(s, a)$ is given for the actions that are different from those of the expert. This algorithm minimizes the difference between the value divergence $w^T F \mu_{\pi^E} - w^T F \mu$ and the policy divergence $l\mu$.

The cost function $c_q$ is convex, but nondifferentiable. Ratliff et al. (2006) showed that $c_q$ can be minimized by using a subgradient method. For a given reward $w$, a subgradient $g_w^q$ is given by:

$$g_w^q = q \left( (w^T F + l)\mu^+ - w^T F \mu_{\pi^E} \right)^{q-1} F \Delta^w \mu_{\pi^E} + \lambda w \tag{3}$$

where $\mu^+ = \arg \max_{\mu \in \mathcal{G}} (w^T F + l)\mu$, and $\Delta^w \mu_{\pi^E} = \mu^+ - \mu_{\pi^E}$.

## 3.3 Linear Programming Apprenticeship Learning

Linear Programming Apprenticeship Learning (LPAL) is based on the following observation: if the reward weights are positive and sum to 1, then $V(\pi) \geqslant V(\pi^E) + \min_i[v_{i,\pi} - v_{i,\pi^E}]$, for any policy $\pi$. LPAL consists in finding a policy that maximizes the margin $\min_i[v_{i,\pi} - v_{i,\pi^E}]$. The maximal margin is found by solving the following linear program:

$$\max_{v, \mu_\pi} \quad v$$

subject to

$$\forall i \in \{0, \dots, k-1\} : v \leqslant \underbrace{\sum_{s \in \mathcal{S}} \sum_{a \in \mathcal{A}} \mu_\pi(s, a) f_i(s, a)}_{v_{i,\pi}} - \underbrace{\sum_{s \in \mathcal{S}} \sum_{a \in \mathcal{A}} \mu_{\pi^E}(s, a) f_i(s, a)}_{v_{i,\pi^E}} \tag{4}$$

$$\mu_\pi(s) = \alpha(s) + \gamma \sum_{s' \in \mathcal{S}} \sum_{a \in \mathcal{A}} \mu_\pi(s', a) T(s', a, s), \sum_{a \in \mathcal{A}} \mu_\pi(s, a) = \mu_\pi(s), \mu_\pi(s, a) \geqslant 0$$

The last three constraints in this linear program correspond to the Bellman-flow constraints (Equation (1)) defining $\mathcal{G}$, the feasible set of $\mu_\pi$. The learned policy $\pi$ is given by:

$$\pi(s, a) = \frac{\mu_\pi(s, a)}{\sum_{a' \in \mathcal{A}} \mu_\pi(s, a')}$$

## 3.4 Approximating feature frequencies

Notice that both MMP and LPAL require the knowledge of the frequencies $v_{i,\pi_{\pi^E}} \overset{def}{=} F(i, .)\mu_{\pi^E}$. These frequencies can be analytically calculated (using Bellman-flow constraints) only if $\pi^E$ is completely specified. Given a sequence of $M$ demonstrated trajectories $t_m = (s_1^m, a_1^m, \dots, s_H^m, a_H^m, )$, the frequencies $v_{i,\pi^E}$ are estimated as:

$$\hat{v}_{i,\pi^E} = \frac{1}{M} \sum_{m=1}^{M} \sum_{t=1}^{H} \gamma^t f_i(s_t^m, a_t^m) \tag{5}$$

There are nevertheless many problems related to this approximation. First, the estimated frequencies $\hat{v}_{i,\pi^E}$ can be very different from the true ones when the demonstration trajectories are scarce. Second, the frequencies $\hat{v}_{i,\pi^E}$ are estimated for a finite horizon $H$, whereas the frequencies $v_{i,\pi}$ used in the objective function (Equations (2) and (4)), are calculated for an infinite horizon (Equation (1)). In practice, these two values are too different and cannot be compared as done in these cost functions. Finally, the frequencies $v_{i,\pi^E}$ are a function of both a policy and the transition probabilities, the empirical estimation of $v_{i,\pi^E}$ does not take advantage of the known transition probabilities.

## 4 Reward loss in Maximum Margin Planning

To show the effect of the error in the estimated feature frequencies on the quality of the learned rewards, we present an analysis of the distance between the vector of reward weights $\hat{w}$ returned by MMP with estimated frequencies $\hat{v}_{\pi^E} = F\hat{\mu}_{\pi^E}$, calculated from the examples by using Equation (5), and the vector $w^E$ returned by MMP with accurate frequencies $v_{\pi^E} = F\mu_{\pi^E}$, calculated by using Equations (1) with the full policy $\pi^E$. We adopt the following notations: $\Delta v_\pi = \hat{v}_{\pi^E} - v_{\pi^E}$, $\Delta w = \hat{w} - w^E$, and $V_l(w) = \max_{\mu \in \mathcal{G}}(w^T F + l)\mu$, and we consider $q = 1$. The following proposition shows how the reward error $\Delta w$ is related to the frequency error $\Delta v_\pi$. Due to the fact

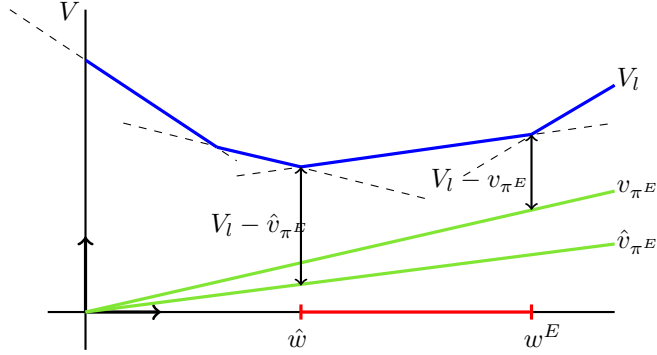

Figure 1: Reward loss in MMP with approximate frequencies $\hat{v}_{\pi^E}$. We indicate by $v_{\pi^E}$ (resp. $\hat{v}_{\pi^E}$) the linear function defined by the vector $v_{\pi^E}$ (resp. $\hat{v}_{\pi^E}$).

that the cost function of MMP is piecewise defined, one cannot find a closed-form relation between $\Delta w$ and $\Delta v_\pi$. However, we show that for any $\hat{w} \in \mathbb{R}^k$, there is a monotonically decreasing function $f$ such that for any $\epsilon \in \mathbb{R}^+$, if $\| \Delta v_\pi \|_2 < f(\epsilon)$ then $\| \Delta w \|_2 \leqslant \epsilon$.

**Proposition 1** *Let $\epsilon \in \mathbb{R}^+$, if $\forall w \in \mathbb{R}^k$, such that $\| w - \hat{w} \|_2 = \epsilon$, if the following condition is verified:*

$$\| \Delta v_\pi \|_2 < \frac{V_l(w) - V_l(\hat{w}) + (\hat{w} - w)^T \hat{v}_{\pi^E} + \frac{\lambda}{2}(\| w \|_2 - \| \hat{w} \|_2)}{\epsilon}$$

*then $\| \Delta w \|_2 \leqslant \epsilon$.*

**Proof** The condition stated in the proposition implies:

$$\| \hat{w} - w \|_2 \| \Delta v_\pi \|_2 < V_l(w) - V_l(\hat{w}) + (\hat{w} - w)^T \hat{v}_{\pi^E} + \frac{\lambda(\| w \|_2 - \| \hat{w} \|_2)}{2}$$

$$\Rightarrow (\hat{w} - w)^T \Delta v_\pi < V_l(w) - V_l(\hat{w}) + (\hat{w} - w)^T \hat{v}_{\pi^E} + \frac{\lambda(\| w \|_2 - \| \hat{w} \|_2)}{2} \quad \text{(Hölder)}$$

$$\Rightarrow V_l(\hat{w}) - \left(\hat{w}^T v_{\pi^E} - \frac{\lambda}{2} \| \hat{w} \|_2\right) < V_l(w) - \left(w^T v_{\pi^E} - \frac{\lambda}{2} \| w \|_2\right)$$

In other terms, the point $(\hat{w}^T v_{\pi^E} - \frac{\lambda}{2} \| \hat{w} \|_2)$ is closer to the surface $V_l$ than any other point $(w^T v_{\pi^E} - \frac{\lambda}{2} \| w \|_2)$, where $w$ is a point on the sphere centered around $\hat{w}$ with a radius of $\epsilon$. Since the function $V_l$ is convex and $(w^{E^T} v_{\pi^E} - \frac{\lambda}{2} \| w^E \|_2)$ is by definition the closest point to the surface $V_l$, then $w^E$ should be inside the ball centered around $\hat{w}$ with a radius of $\epsilon$. Therefore, $\| w^E - \hat{w} \|_2 \leqslant \epsilon$ and thus $\| \Delta w \|_2 \leqslant \epsilon$. $\square$

Consequently, the reward loss $\| \Delta w \|_2$ approaches zero as the error of the estimated feature frequencies $\| \Delta v_\pi \|_2$ approaches zero. A simpler bound can be easily derived given admissible heuristics of $V_l$.

**Corollary**: Let $\underline{V_l}$ and $\overline{V_l}$ be respectively a lower and an upper bound on $V_l$, then Proposition (1) holds if $V_l(w) - V_l(\hat{w})$ is replaced by $\underline{V_l}(w) - \overline{V_l}(\hat{w})$.

Figure (1) illustrates the divergence from the optimal reward weight $w^E$ when approximate frequencies are used. The error is not a continuous function of $\Delta v_\pi$ when the cost function is not regularized, because the vector returned by MMP is always a fringe point. Informally, the error is proportional to the maximum subgradient of the function $V_l - v_{\pi^E}$ at the fringe point $w^E$.

# 5 Bootstrapping Maximum Margin Planning

The feature frequency error $\Delta v_\pi$ can be significantly reduced by using the known transition function for calculating $\hat{v}_{\pi^E}$ and solving the flow Equations (1), instead of the Monte Carlo estimator (Equation (5)). However, this cannot be done unless the complete expert's policy $\pi^E$ is provided.

Assuming that the expert's policy $\pi^E$ is optimal and deterministic, the value $w^T F \mu_{\pi^E}$ in Equation (2) can be replaced by $\max_{\mu \in \mathcal{G}_{\pi_E}} w^T F \mu$, the value of the optimal policy, according to the current reward weight $w$, that selects the same actions as the expert in all the states that occurred in the demonstration. The cost function of the *bootstrapped* Maximum Margin Planning becomes:

$$c_q(w) = \left( \max_{\mu_1 \in \mathcal{G}} (w^T F + l) \mu_1 - \max_{\mu_2 \in \mathcal{G}_{\pi_E}} w^T F \mu_2 \right)^q + \frac{\lambda}{2} \| w \|^2 \qquad (6)$$

where $\mathcal{G}_{\pi^E}$ is the set of vectors $\mu_\pi$, subject to the following modified Bellman-flow constraints:

$$\mu_\pi(s) = \alpha(s) + \gamma \sum_{s' \in \mathcal{S}_e} \mu_\pi(s') \sum_{a \in \mathcal{A}} \pi^E(s', a) T^a(s', s) + \gamma \sum_{s' \in \mathcal{S} \setminus \mathcal{S}_e} \sum_{a \in \mathcal{A}} \mu_\pi(s', a) T^a(s', s)$$

$$\sum_{a \in \mathcal{A}} \mu_\pi(s, a) = \mu_\pi(s), \mu_\pi(s, a) \geqslant 0 \qquad (7)$$

$\mathcal{S}_e$ is the set of states encountered in the demonstrations, where the expert's policy is known.

Unfortunately, the new cost function (Equation (6)) is not necessarily convex. In fact, it corresponds to a margin between two convex functions: the value of the bootstrapped expert's policy $\max_{\mu \in \mathcal{G}_{\pi_E}} w^T F \mu$ and the value of the best alternative policy $\max_{\mu \in \mathcal{G}} (w^T F + l)\mu$. Yet, a local optimal solution of this modified cost function can be found by using the same subgradient as in Equation (3), and replacing $\mu_{\pi^E}$ by $\arg\max_{\mu \in \mathcal{G}_{\pi_E}} w^T F \mu$. In practice, as we will show in the experimental analysis, the solution returned by the bootstrapped MMP outperforms the solution of MMP where the expert's frequency is calculated without taking into account the known transition probabilities. This improvement is particularly pronounced in highly stochastic environments. The computational cost of minimizing this modified cost function is twice the one of MMP, since two optimal policies are found at each iteration.

In the remainder of this section, we provide a theoretical analysis of the cost function given by Equation (6). For the sake of simplicity, we consider $q = 1$ and $\lambda = 0$.

**Proposition 2** *The cost function defined by Equation (6), has at most $\frac{|\mathcal{A}|^{|\mathcal{S}|}}{|\mathcal{A}|^{|\mathcal{S}_e|}}$ different local minima.*

**Proof** If $q = 1$ and $\lambda = 0$, then the cost $c_q(w)$ corresponds to a distance between the convex and piecewise linear functions $\max_{\mu \in \mathcal{G}} (w^T F + l)\mu$ and $\max_{\mu \in \mathcal{G}_{\pi_E}} w^T F \mu$. Therefore, for any vector $\mu' \in \mathcal{G}_{\pi^E}$, the function $c_q$ is monotone in the interval of $w$ where $\mu'$ is optimal, i.e. where $w^T F \mu' = \max_{\mu \in \mathcal{G}_{\pi_E}} w^T F \mu$. Consequently, the number of local minima of the function $c_q$ is at most equal to the number of optimal vectors $\mu$ in $\mathcal{G}_{\pi^E}$, which is upper bounded by the number of deterministic policies defined on $S \setminus \mathcal{S}_e$, i.e. by $|\mathcal{A}|^{|\mathcal{S}| - |\mathcal{S}_e|}$. $\square$

Consequently, the number of different local minima of the function $c_q$ decreases as the number of states covered by the demonstration increases. Ultimately, the function $c_q$ becomes convex when the demonstration covers all the possible states.

**Theorem 1** *If there exists a reward weight vector $w^* \in \mathbb{R}^k$, such that the expert's policy $\pi^E$ is the only optimal policy with $w^*$, i.e. $\arg\max_{\mu \in \mathcal{G}} w^{*T} F \mu = \{\mu_{\pi^E}\}$, then there exists $\alpha > 0$ such that: (i), the expert's policy $\pi^E$ is the only optimal policy with $\alpha w^*$, and (ii), $c_q(\alpha w^*)$ is a local minimum of the function $c_q$, defined in Equation (6).*

**Proof** The set of subgradients of function $c_q$ at a point $w \in \mathbb{R}^k$, denoted by $\nabla_w c_q(w)$, corresponds to vectors $F\mu' - F\mu''$, with $\mu' \in \arg\max_{\mu \in \mathcal{G}} (w^T F + l)\mu$ and $\mu'' \in \arg\max_{\mu \in \mathcal{G}_{\pi_E}} w^T F \mu$. In order that $c_q(w)$ will be a local minimum, it suffices to ensure that $\vec{0} \in \nabla_w c_q(w)$, i.e. $\exists \mu' \in \arg\max_{\mu \in \mathcal{G}} (w^T F + l)\mu, \exists \mu'' \in \arg\max_{\mu \in \mathcal{G}_{\pi_E}} w^T F \mu$ such that $F\mu' = F\mu''$. Let $w^* \in \mathbb{R}^k$

be a reward weight vector such that $\pi^E$ is the only optimal policy, and let $\epsilon = w^{*T}F\mu_{\pi^E} - w^{*T}F\mu'$ where $\mu' \in \arg\max_{\mu \in \mathcal{G}-\{\mu_{\pi^E}\}} w^{*T}F\mu$. Then, $\alpha w^{*T}F\mu_{\pi^E} - \alpha w^{*T}F\mu' = \frac{2|\mathcal{S}_e|}{1-\gamma}$, where $\alpha = \frac{2|\mathcal{S}_e|}{\epsilon(1-\gamma)}$. Notice that by multiplying $w^*$ by $\alpha > 0$, $\pi^E$ remains the only optimal policy, i.e. $\arg\max_{\mu \in \mathcal{G}} \alpha w^{*T}F\mu = \{\mu_{\pi^E}\}$, and $\mu' \in \arg\max_{\mu \in \mathcal{G}-\{\mu_{\pi^E}\}} \alpha w^{*T}F\mu$. Therefore, it suffices to show that $\mu_{\pi^E} \in \arg\max_{\mu \in \mathcal{G}}(\alpha w^{*T}F + l)\mu$. Indeed, $\max_{\mu \in \mathcal{G}-\{\pi^E\}}(\alpha w^{*T}F + l)\mu \leqslant \max_{\mu \in \mathcal{G}-\{\pi^E\}} \alpha w^{*T}F\mu + \max_{\mu \in \mathcal{G}-\{\pi^E\}} l\mu \leqslant \left(\alpha w^{*T}F\mu_{\pi^E} - \frac{2|\mathcal{S}_e|}{1-\gamma}\right) + \frac{|\mathcal{S}_e|}{1-\gamma} \leqslant \alpha w^{*T}F\mu_{\pi^E} - \frac{|\mathcal{S}_e|}{1-\gamma}$, therefore, $\mu_{\pi^E} \in \arg\max_{\mu \in \mathcal{G}}(\alpha w^{*T}F + l)\mu.\square$

# 6 Bootstrapping Linear Programming Apprenticeship Learning

As with MMP, the feature frequencies in LPAL can be analytically calculated only when a complete policy $\pi^E$ of the expert is provided. Alternatively, the same error bound $V(\pi) \geqslant V(\pi^E) + v$ can be guaranteed by setting $v = \min_{i=0,\dots,k-1} \min_{\pi' \in \Pi^E}[v_{i,\pi} - v_{i,\pi'}]$, where $\Pi^E$ denotes the set of all the policies that select the same actions as the expert in all the states that occurred in the demonstration, assuming $\pi^E$ is deterministic (In LPAL, $\pi^E$ is not necessarily an optimal policy). Instead of enumerating all the policies of the set $\Pi^E$ in the constraints, note that $v = \min_{i=0,\dots,k-1}[v_{i,\pi} - v_i^E]$, where $v_i^E \stackrel{def}{=} \max_{\pi' \in \Pi^E} v_{i,\pi'}$ for each feature $i$. Therefore, LPAL can be reformulated as maximizing the margin $\min_{i=0,\dots,k-1}[v_{i,\pi} - v_i^E]$.

The maximal margin is found by solving the following linear program:

$$\max_{v,\mu_\pi} \quad v$$

subject to

$$\forall i \in \{0,\dots,k-1\} : v \leqslant \underbrace{\sum_{s \in \mathcal{S}} \sum_{a \in \mathcal{A}} \mu_\pi(s,a)f_i(s,a)}_{v_{i,\pi}} - \underbrace{\sum_{s \in \mathcal{S}} \sum_{a \in \mathcal{A}} \mu_{i,\pi'}(s,a)f_i(s,a)}_{v_i^E}$$

$$\mu_\pi(s) = \alpha(s) + \gamma \sum_{s' \in \mathcal{S}} \sum_{a \in \mathcal{A}} \mu_\pi(s',a)T(s',a,s), \sum_{a \in \mathcal{A}} \mu_\pi(s,a) = \mu_\pi(s), \mu_\pi(s,a) \geqslant 0$$

where the values $v_i^E$ are found by solving $k$ separate optimization problems ($k$ is the number of features). For each feature $i$, $v_i^E$ is the value of the optimal policy in the set $\Pi^E$ under the reward weights $w$ defined as: $w_i = 1$ and $w_j = 0, \forall j \neq i$.

# 7 Experimental Results

To validate our approach, we experimented on two simulated navigation problems: a gridworld and two racetrack domains, taken from (Boularias & Chaib-draa, 2010). While these are not meant to be challenging tasks, they allow us to compare our approach to other methods of apprenticeship learning, namely MMP and LPAL with Monte Carlo estimation, and a simple classification algorithm where the action in a given state is selected by performing a majority vote on the $k$-nearest neighbor states where the expert's action is known. For each state, the distance $k$ is gradually increased until at least one known state is encountered. The distance between two states corresponds to the shortest path between them with a positive probability.

## 7.1 Gridworld

We consider $16 \times 16$ and $24 \times 24$ gridworlds. The state corresponds to the location of the agent on the grid. The agent has four actions for moving in one of the four directions of the compass. The actions succeed with probability $0.9$. The gridworld is divided into non-overlapping regions, and the reward varies depending on the region in which the agent is located. For each region $i$, there is a feature $f_i$, where $f_i(s)$ indicates whether state $s$ is in region $i$. The expert's policy $\pi^E$ corresponds to the optimal deterministic policy found by value iteration. In all our experiments on gridworlds, we used only 10 demonstration trajectories, which is a significantly small number compared to other methods ( Neu & Szepesvri (2007) for example). The duration of the trajectories is 50 time-steps.

| Size | Features | Expert | k-NN | MMP + MC | MMP + Bootstrap | LPAL + MC | LPAL + Bootstrap |
|---|---|---|---|---|---|---|---|
| $16 \times 16$ | 16 | 0.4672 | 0.4635 | 0.0000 | **0.4678** | 0.0380 | **0.1572** |
| $16 \times 16$ | 64 | 0.5281 | 0.5198 | 0.0000 | **0.5252** | 0.0255 | **0.4351** |
| $16 \times 16$ | 256 | 0.3988 | 0.4062 | 0.0537 | **0.3828** | 0.0555 | **0.1706** |
| | | | | | | | |
| $24 \times 24$ | 64 | 0.5210 | 0.6334 | 0.0000 | **0.5217** | 0.0149 | **0.2767** |
| $24 \times 24$ | 144 | 0.5916 | 0.5876 | 0.0122 | **0.5252** | 0.0400 | **0.4432** |
| $24 \times 24$ | 576 | 0.3102 | 0.2814 | 0.0974 | **0.0514** | 0.0439 | **0.0349** |

Table 1: Gridworld average reward results

Table 1 shows the average reward per step of the learned policy, averaged over $10^3$ independent trials of the same duration as the demonstration trajectories. Our first observation is that Bootstrapped MMP learned policies just as good as the expert's policy, while both MMP and LPAL using Monte Carlo (MC) estimator remarkably failed to collect any reward. This is due to the fact that we used a very small number of demonstrations ($10 \times 50$ time-steps) compared to the size of these problems. Note that this problem is not specific to MMP or LPAL. In fact, any other algorithm using the same approximation method would produce similar results. The second observation is that the values of the policies learned by bootstrapped LPAL were between the values of LPAL with Monte Carlo and the optimal ones. In fact, the policy learned by the bootstrapped LPAL is one that minimizes the difference between the expected frequency of a feature using this policy and the maximal one among all the policies that resemble to the expert's policy. Therefore, the learned policy maximizes the frequency of a feature that is not necessary a good one (with a high reward weight). We also notice that the performance of all the tested algorithms was low when $576$ features were used. In this case, every feature takes a non null weight in one state only. Therefore, the demonstrations did not provide enough information about the rewards of the states that were not visited by the expert. Finally, we remark that $k$-NN performed as an expert in this experiment. In fact, since there are no obstacles on the grid, neighboring states often have similar optimal actions.

## 7.2 Racetrack

We implemented a simplified car race simulator, a detailed description of the corresponding race-tracks was provided in (Boularias & Chaib-draa, 2010). The states correspond to the position of the car on the racetrack and its velocity. For racetrack (1), the car always starts from the same initial position, and the duration of each demonstration trajectory is 20 time-steps. For racetrack (2), the car starts at a random position, and the length of each trajectory is 40 time-steps. A high reward is given for reaching the finish line, a low cost is associated to each movement, and high cost is associated to driving off-road (or hitting an obstacle). Figure 2 (a-f) shows the average reward per step of the learned policies, the average proportion of off-road steps, and the average number of steps before reaching the finish line, as a function of the number of trajectories in the demonstration. We first notice that $k$-NN performed poorly, this is principally caused by the effect of driving off-road on both the cumulated reward and the velocity of the car. In this context, neighbor states do not necessarily share the same optimal action. Contrary to the gridworld experiments, MMP with Monte Carlo achieved good performances on racetrack (1). In fact, by fixing the initial state, the demonstration covers most of the reachable states, and the feature frequencies are accurately estimated from the demonstration. On racetrack (2) however, MMP with MC was unable to learn a good policy because all the states were reachable from the initial distribution. Similarly, LPAL with both MC and bootstrapping failed to achieve good results on racetracks (1) and (2). This is due to the fact that LPAL tries to maximize the frequency of features that are not necessary associated to a high reward, such as hitting obstacles. Finally, we notice the nearly optimal performance of the bootstrapped MMP, on both racetracks (1) and (2).

## 8 Conclusion and Future Work

The main question of apprenticeship learning is how to generalize the expert's policy to states that have not been encountered during the demonstration. Inverse Reinforcement Learning (IRL) provides an efficient answer which consists in first learning a reward function that explains the observed behavior, and then using it for the generalization. A strong assumption considered in IRL-based al-

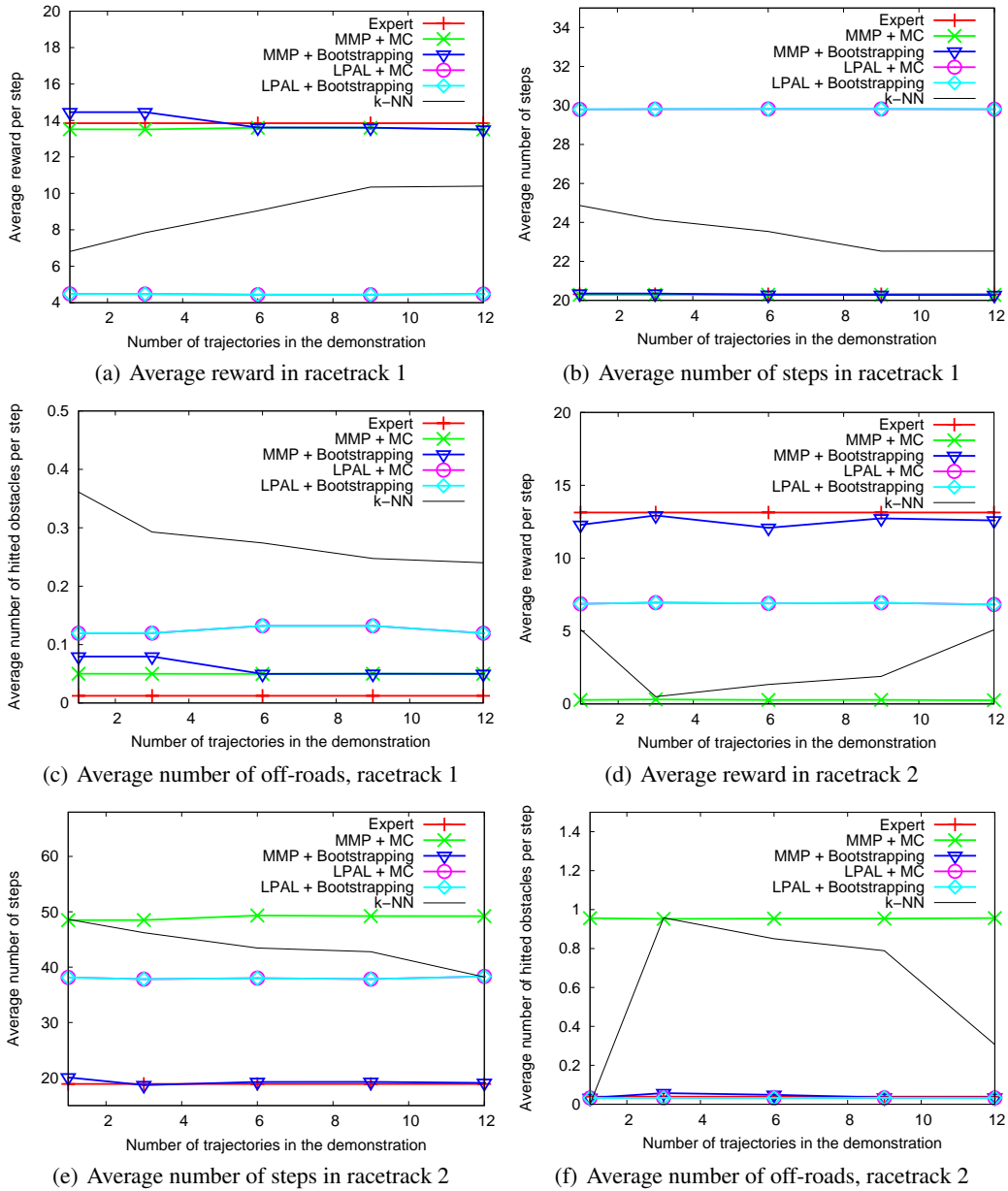

Figure 2: Racetrack results

gorithms is that the reward is a linear function of state-action features, and the frequencies of these features can be estimated from a few demonstrations even if these demonstrations cover only a small part of the state space. In this paper, we showed that this assumption does not hold in highly stochastic systems. We also showed that this problem can be solved by modifying the cost function so that the value of the learned policy is compared to the exact value of a generalized expert's policy. We also provided theoretical insights on the modified cost function, showing that it admits the expert's true reward as a locally optimal solution, under mild conditions. The empirical analysis confirmed the outperformance of Bootstrapped MMP in particular. These promising results push us to further investigate the theoretical properties of the modified cost function.

As a future work, we mainly target to compare this approach with the one proposed by Ratliff et al. (2007), where the base features are boosted by using a classifier.

# References

Abbeel, Pieter and Ng, Andrew Y. Apprenticeship Learning via Inverse Reinforcement Learning. In *Proceedings of the Twenty-first International Conference on Machine Learning (ICML'04)*, pp. 1–8, 2004.

Boularias, Abdeslam and Chaib-draa, Brahim. Apprenticeship Learning via Soft Local Homomorphisms. In *Proceedings of 2010 IEEE International Conference on Robotics and Automation (ICRA'10)*, pp. 2971–2976, 2010.

Neu, Gergely and Szepesvri, Csaba. Apprenticeship Learning using Inverse Reinforcement Learning and Gradient Methods. In *Conference on Uncertainty in Artificial Intelligence (UAI'07)*, pp. 295–302, 2007.

Ng, Andrew and Russell, Stuart. Algorithms for Inverse Reinforcement Learning. In *Proceedings of the Seventeenth International Conference on Machine Learning (ICML'00)*, pp. 663–670, 2000.

Ramachandran, Deepak and Amir, Eyal. Bayesian Inverse Reinforcement Learning. In *Proceedings of The twentieth International Joint Conference on Artificial Intelligence (IJCAI'07)*, pp. 2586–2591, 2007.

Ratliff, N., Bagnell, J., and Zinkevich, M. Maximum Margin Planning. In *Proceedings of the Twenty-third International Conference on Machine Learning (ICML'06)*, pp. 729–736, 2006.

Ratliff, Nathan, Bradley, David, Bagnell, J. Andrew, and Chestnutt, Joel. Boosting Structured Prediction for Imitation Learning. In *Advances in Neural Information Processing Systems 19 (NIPS'07)*, pp. 1153–1160, 2007.

Syed, Umar and Schapire, Robert. A Game-Theoretic Approach to Apprenticeship Learning. In *Advances in Neural Information Processing Systems 20 (NIPS'08)*, pp. 1449–1456, 2008.

Syed, Umar, Bowling, Michael, and Schapire, Robert E. Apprenticeship Learning using Linear Programming. In *Proceedings of the Twenty-fifth International Conference on Machine Learning (ICML'08)*, pp. 1032–1039, 2008.

